# Policy search by dynamic programming

**J. Andrew Bagnell**
Carnegie Mellon University
Pittsburgh, PA 15213

**Sham Kakade**
University of Pennsylvania
Philadelphia, PA 19104

**Andrew Y. Ng**
Stanford University
Stanford, CA 94305

**Jeff Schneider**
Carnegie Mellon University
Pittsburgh, PA 15213

## Abstract

We consider the policy search approach to reinforcement learning. We show that if a "baseline distribution" is given (indicating roughly how often we expect a good policy to visit each state), then we can derive a policy search algorithm that terminates in a finite number of steps, and for which we can provide non-trivial performance guarantees. We also demonstrate this algorithm on several grid-world POMDPs, a planar biped walking robot, and a double-pole balancing problem.

## 1 Introduction

Policy search approaches to reinforcement learning represent a promising method for solving POMDPs and large MDPs. In the policy search setting, we assume that we are given some class $\Pi$ of policies mapping from the states to the actions, and wish to find a good policy $\pi \in \Pi$. A common problem with policy search is that the search through $\Pi$ can be difficult and computationally expensive, and is thus typically based on local search heuristics that do not come with any performance guarantees.

In this paper, we show that if we give the learning agent a "base distribution" on states (specifically, one that indicates how often we expect it to be in each state; cf. [5, 4]), then we can derive an efficient policy search algorithm that terminates after a polynomial number of steps. Our algorithm outputs a non-stationary policy, and each step in the algorithm requires only a minimization that can be performed or approximated via a call to a standard supervised learning algorithm. We also provide non-trivial guarantees on the quality of the policies found, and demonstrate the algorithm on several problems.

## 2 Preliminaries

We consider an MDP with state space $S$; initial state $s_0 \in S$; action space $A$; state transition probabilities $\{P_{sa}(\cdot)\}$ (here, $P_{sa}$ is the next-state distribution on taking action $a$ in state $s$); and reward function $R : S \mapsto \mathbb{R}$, which we assume to be bounded in the interval $[0, 1]$.

In the setting in which the goal is to optimize the sum of discounted rewards over an infinite-horizon, it is well known that an optimal policy which is both Markov and stationary (i.e., one where the action taken does not depend on the current time) always exists. For this reason, learning approaches to infinite-horizon discounted MDPs have typically focused

on searching for stationary policies (e.g., [8, 5, 9]). In this work, we consider policy search in the space of non-starionary policies, and show how, with a base distribution, this allows us to derive an efficient algorithm.

We consider a setting in which the goal is to maximize the sum of undiscounted rewards over a $T$ step horizon: $\frac{1}{T}\mathrm{E}[R(s_0) + R(s_1) + \ldots + R(s_{T-1})]$. Clearly, by choosing $T$ sufficiently large, a finite-horizon problem can also be used to approximate arbitrarily well an infinite-horizon discounted problem. (E.g., [6]) Given a *non-stationary* policy $(\pi_t, \pi_{t+1}, \ldots, \pi_{T-1})$, where each $\pi_t : S \mapsto A$ is a (stationary) policy, we define the value

$$V_{\pi_t,\ldots,\pi_{T-1}}(s) \equiv \frac{1}{T}\mathrm{E}[R(s_t) + R(s_{t+1}) + \ldots + R(s_{T-1})|s_t = s; (\pi_t, \ldots, \pi_{T-1})]$$

as the expected (normalized) sum of rewards attained by starting at state $s$ and the "clock" at time $t$, taking one action according to $\pi_t$, taking the next action according to $\pi_{t+1}$, and so on. Note that

$$V_{\pi_t,\ldots,\pi_{T-1}}(s) \equiv \frac{1}{T}R(s) + \mathrm{E}_{s'\sim P_{s\pi_t(s)}}[V_{\pi_{t+1},\ldots,\pi_{T-1}}(s)],$$

where the "$s' \sim P_{s\pi_t(s)}$" subscript indicates that the expectation is with respect to $s'$ drawn from the state transition distribution $P_{s\pi_t(s)}$.

In our policy search setting, we consider a restricted class of deterministic, stationary policies $\Pi$, where each $\pi \in \Pi$ is a map $\pi : S \mapsto A$, and a corresponding class of *non-stationary* policies $\Pi^T = \{(\pi_0, \pi_1, \ldots, \pi_{T-1}) \mid$ for all t, $\pi_t \in \Pi\}$. In the partially observed, POMDP setting, we may restrict $\Pi$ to contain policies that depend only on the observable aspects of the state, in which case we obtain a class of memoryless/reactive policies. Our goal is to find a non-stationary policy $(\pi_0, \pi_1 \ldots, \pi_{T-1}) \in \Pi^T$ which performs well under the performance measure $V_{\pi_0,\pi_1\ldots,\pi_{T-1}}(s_0)$, which we abbreviate as $V_\pi(s_0)$ when there is no risk of confusion.

## 3 The Policy Search Algorithm

Following [5, 4], we assume that we are given a sequence of base distributions $\mu_0, \mu_1, \ldots, \mu_{T-1}$ over the states. Informally, we think of $\mu_t$ as indicating to the algorithm approximately how often we think a good policy visits each state at time $t$.

Our algorithm (also given in [4]), which we call Policy Search by Dynamic Programming (PSDP) is in the spirit of the traditional dynamic programming approach to solving MDPs where values are "backed up." In PSDP, it is the policy which is backed up. The algorithm begins by finding $\pi_{T-1}$, then $\pi_{T-2}, \ldots$ down to $\pi_0$. Each policy $\pi_t$ is chosen from the stationary policy class $\Pi$. More formally, the algorithm is as follows:

**Algorithm 1 (PSDP)** *Given $T$, $\mu_t$, and $\Pi$:*

> *for $t = T - 1, T - 2, \ldots, 0$*
>
> > *Set $\pi_t = \arg\max_{\pi'\in\Pi}\mathrm{E}_{s\sim\mu_t}[V_{\pi',\pi_{t+1}\ldots,\pi_{T-1}}(s)]$*

In other words, we choose $\pi_t$ from $\Pi$ so as to maximize the expected sum of future rewards for executing actions according to the policy sequence $(\pi_t, \pi_{t+1}, \ldots, \pi_{T-1})$ when starting from a random initial state $s$ drawn from the baseline distribution $\mu_t$.

Since $\mu_0, \ldots, \mu_{T-1}$ provides the distribution over the state space that the algorithm is optimizing with respect to, we might hope that if a good policy tends to visit the state space in a manner comparable to this base distribution, then PSDP will return a good policy. The following theorem formalizes this intuition. The theorem also allows for the situation where the maximization step in the algorithm (the $\arg\max_{\pi'\in\Pi}$) can be done only approximately. We later give specific examples showing settings in which this maximization can (approximately or exactly) be done efficiently.

The following definitions will be useful. For a non-stationary policy $\pi = (\pi_0, \ldots, \pi_{T-1})$, define the future state distribution

$$\mu_{\pi,t}(s) = \Pr(s_t = s|s_0, \pi).$$

I.e. $\mu_{\pi,t}(s)$ is the probability that we will be in state $s$ at time $t$ if picking actions according to $\pi$ and starting from state $s_0$. Also, given two $T$-step sequences of distributions over states $\mu = (\mu_0, \ldots, \mu_t)$ and $\mu' = (\mu'_0, \ldots, \mu'_t)$, define the average **variational distance** between them to be[1]

$$d_{\mathrm{var}}(\mu, \mu') \equiv \frac{1}{T} \sum_{t=0}^{T-1} \sum_{s \in S} |\mu_t(s) - \mu'_t(s)|$$

Hence, if $\pi_{\mathrm{ref}}$ is some policy, then $d_{\mathrm{var}}(\mu, \mu_{\pi_{\mathrm{ref}}})$ represents how much the base distribution $\mu$ differs from the future state distribution of the policy $\pi_{\mathrm{ref}}$.

**Theorem 1 (Performance Guarantee)** *Let $\pi = (\pi_0, \ldots, \pi_{T-1})$ be a non-stationary policy returned by an $\varepsilon$-approximate version of PSDP in which, on each step, the policy $\pi_t$ found comes within $\varepsilon$ of maximizing the value. I.e.,*

$$\mathrm{E}_{s \sim \mu_t}[V_{\pi_t, \pi_{t+1} \ldots, \pi_{T-1}}(s)] \geq \max_{\pi' \in \Pi} \mathrm{E}_{s \sim \mu_t}[V_{\pi', \pi_{t+1} \ldots, \pi_{T-1}}(s)] - \varepsilon. \qquad (1)$$

*Then for all $\pi_{\mathrm{ref}} \in \Pi^T$ we have that*

$$V_\pi(s_0) \geq V_{\pi_{\mathrm{ref}}}(s_0) - T\varepsilon - T d_{\mathrm{var}}(\mu, \mu_{\pi_{\mathrm{ref}}}).$$

**Proof.** This proof may also be found in [4], but for the sake of completeness, we also provide it here. Let $P_t(s) = \Pr(s_t = s | s_0, \pi_{\mathrm{ref}})$, $\pi_{\mathrm{ref}} = (\pi_{\mathrm{ref},0}, \ldots, \pi_{\mathrm{ref},T-1}) \in \Pi^T$, and $\pi = (\pi_0, \ldots, \pi_{T-1})$ be the output of $\varepsilon$-PSDP. We have

$$
\begin{aligned}
V_{\pi_{\mathrm{ref}}} - V_\pi &= \frac{1}{T} \sum_{t=0}^{T-1} \mathrm{E}_{s_t \sim P_t}[R(s_t)] - V_{\pi_0, \ldots}(s) \\
&= \sum_{t=0}^{T-1} \mathrm{E}_{s_t \sim P_t}[\frac{1}{T} R(s_t) + V_{\pi_t, \ldots}(s_t) - V_{\pi_t, \ldots}(s_t)] - V_{\pi_0, \ldots}(s) \\
&= \sum_{t=0}^{T-1} \mathrm{E}_{s_t \sim P_t, s_{t+1} \sim P_{s_t \pi_{\mathrm{ref},t}(s_t)}}[\frac{1}{T} R(s_t) + V_{\pi_{t+1}, \ldots}(s_{t+1}) - V_{\pi_t, \ldots}(s_t)] \\
&= \sum_{t=0}^{T-1} \mathrm{E}_{s_t \sim P_t}[V_{\pi_{\mathrm{ref},t}, \pi_{t+1}, \ldots, \pi_{T-1}}(s_t) - V_{\pi_t, \pi_{t+1}, \ldots, \pi_{T-1}}(s_t)]
\end{aligned}
$$

It is well-known that for any function $f$ bounded in absolute value by $B$, it holds true that $|E_{s \sim \mu_1}[f(s)] - E_{s \sim \mu_2}[f(s)]| \leq B \sum_s |\mu_1(s) - \mu_2(s)|$. Since the values are bounded in the interval $[0, 1]$ and since $P_t = \mu_{\pi_{\mathrm{ref}},t}$,

$$
\begin{aligned}
&\sum_{t=0}^{T-1} \mathrm{E}_{s_t \sim P_t}[V_{\pi_{\mathrm{ref},t}, \pi_{t+1}, \ldots, \pi_{T-1}}(s_t) - V_{\pi_t, \pi_{t+1}, \ldots, \pi_{T-1}}(s_t)] \\
&\leq \sum_{t=0}^{T-1} \mathrm{E}_{s \sim \mu_t}[V_{\pi_{\mathrm{ref},t}, \pi_{t+1}, \ldots, \pi_{T-1}}(s) - V_{\pi_t, \pi_{t+1}, \ldots, \pi_{T-1}}(s)] - \sum_{t=0}^{T-1} |P_t(s) - \mu_t(s)| \\
&\leq \sum_{t=0}^{T-1} \max_{\pi' \in \Pi} \mathrm{E}_{s \sim \mu_t}[V_{\pi', \pi_{t+1}, \ldots, \pi_{T-1}}(s) - V_{\pi_t, \pi_{t+1}, \ldots, \pi_{T-1}}(s)] - T d_{\mathrm{var}}(\mu_{\pi_{\mathrm{ref}}}, \mu) \\
&\leq T\varepsilon + T d_{\mathrm{var}}(\mu_{\pi_{\mathrm{ref}}}, \mu)
\end{aligned}
$$

where we have used equation (1) and the fact that $\pi_{\mathrm{ref}} \in \Pi^T$. The result now follows. $\quad\square$

This theorem shows that PSDP returns a policy with performance that competes favorably against those policies $\pi_{\mathrm{ref}}$ in $\Pi^T$ whose future state distributions are close to $\mu$. Hence, we expect our algorithm to provide a good policy if our prior knowledge allows us to choose a $\mu$ that is close to a future state distribution for a good policy in $\Pi^T$.

It is also shown in [4] that the dependence on $d_{\mathrm{var}}$ is tight in the worst case. Furthermore, it is straightforward to show (cf. [6, 8]) that the $\varepsilon$-approximate PSDP can be implemented using a number of samples that is linear in the VC dimension of $\Pi$, polynomial in $T$ and $\frac{1}{\varepsilon}$, but otherwise independent of the size of the state space. (See [4] for details.)

## 4  Instantiations

In this section, we provide detailed examples showing how PSDP may be applied to specific classes of policies, where we can demonstrate *computational* efficiency.

## 4.1 Discrete observation POMDPs

Finding memoryless policies for POMDPs represents a difficult and important problem. Further, it is known that the best memoryless, stochastic, stationary policy can perform better by an arbitrarily large amount than the best memoryless, deterministic policy. This is frequently given as a reason for using stochastic policies. However, as we shortly show, there is no advantage to using stochastic (rather than deterministic) policies, when we are searching for non-stationary policies.

Four natural classes of memoryless policies to consider are as follows: stationary deterministic (*SD*), stationary stochastic (*SS*), non-stationary deterministic (*ND*) and non-stationary stochastic (*NS*). Let the operator opt return the value of the optimal policy in a class. The following specifies the relations among these classes.

**Proposition 1 (Policy ordering)** *For any finite-state, finite-action POMDP,*
$$opt(SD) \leq opt(SS) \leq opt(ND) = opt(NS)$$

We now sketch a proof of this result. To see that $\mathrm{opt}(ND) = \mathrm{opt}(NS)$, let $\mu_{NS}$ be the future distribution of an optimal policy $\pi_{NS} \in NS$. Consider running PSDP with base distribution $\mu_{NS}$. After each update, the resulting policy $(\pi_{NS,0}, \pi_{NS,1}, \ldots, \pi_t, \ldots, \pi_T)$ must be at least as good as $\pi_{NS}$. Essentially, we can consider PSDP as sweeping through each timestep and modifying the stochastic policy to be deterministic, while never decreasing performance. A similar argument shows that $\mathrm{opt}(SS) \leq \mathrm{opt}(ND)$ while a simple example POMDP in the next section demonstrates this inequality can be strict.

The potentially superior performance of non-stationary policies contrasted with stationary stochastic ones provides further justification for their use. Furthermore, the last inequality suggests that only considering deterministic policies is sufficient in the non-stationary regime.

Unfortunately, one can show that it is NP-hard to exactly or approximately find the best policy in any of these classes (this was shown for *SD* in [7]). While many search heuristics have been proposed, we now show PSDP offers a viable, computationally tractable, alternative for finding a good policy for POMDPs, one which offers performance guarantees in the form of Theorem 1.

**Proposition 2 (PSDP complexity)** *For any POMDP, exact PSDP ($\varepsilon = 0$) runs in time polynomial in the size of the state and observation spaces and in the horizon time $T$.*

Under PSDP, the policy update is as follows:
$$\pi_t(o) = \arg\max_a \mathrm{E}_{s \sim \mu_t}[p(o|s) V_{a,\pi_{t+1}\ldots,\pi_{T-1}}(s)], \tag{2}$$
where $p(o|s)$ is the observation probabilities of the POMDP and the policy sequence $(a, \pi_{t+1} \ldots, \pi_{T-1})$ always begins by taking action $a$. It is clear that given the policies from time $t+1$ onwards, $V_{a,\pi_{t+1}\ldots,\pi_{T-1}}(s)$ can be efficiently computed and thus the update 2 can be performed in polynomial time in the relevant quantities. Intuitively, the distribution $\mu$ specifies here how to trade-off the benefits of different underlying state-action pairs that share an observation. Ideally, it is the distribution provided by an optimal policy for *ND* that optimally specifies this tradeoff.

This result does not contradict the NP-hardness results, because it requires that a good baseline distribution $\mu$ be provided to the algorithm. However, if $\mu$ is the future state distribution of the optimal policy in *ND*, then PSDP returns an optimal policy for this class in polynomial time.

Furthermore, if the state space is prohibitively large to perform the exact update in equation 2, then Monte Carlo integration may be used to evaluate the expectation over the state space. This leads to an $\varepsilon$-approximate version of PSDP, where one can obtain an algorithm with *no dependence* on the size of the state space and a polynomial dependence on the number of observations, $T$, and $\frac{1}{\varepsilon}$ (see discussion in [4]).

### 4.2 Action-value approximation

PSDP can also be efficiently implemented if it is possible to efficiently find an approximate action-value function $\tilde{V}_{a,\pi_{t+1}...,\pi_{T-1}}(s)$, i.e., if at each timestep

$$\epsilon \geq \mathrm{E}_{s \sim \mu_t}[\max_{a \in A} |\tilde{V}_{a,\pi_{t+1}...,\pi_{T-1}}(s) - V_{a,\pi_{t+1}...,\pi_{T-1}}(s)|].$$

(Recall that the policy sequence $(a, \pi_{t+1} \ldots, \pi_{T-1})$ always begins by taking action $a$.) If the policy $\pi_t$ is greedy with respect to the action value $\tilde{V}_{a,\pi_{t+1}...,\pi_{T-1}}(s)$ then it follows immediately from Theorem 1 that our policy value differs from the optimal one by $2T\epsilon$ plus the $\mu$ dependent variational penalty term. It is important to note that this error is phrased in terms of an average error over state-space, as opposed to the worst case errors over the state space that are more standard in RL. We can intuitively grasp this by observing that value iteration style algorithms may amplify any small error in the value function by pushing more probability mass through where these errors are. PSDP, however, as it does not use value function backups, cannot make this same error; the use of the computed policies in the future keeps it honest. There are numerous efficient regression algorithms that can minimize this, or approximations to it.

### 4.3 Linear policy MDPs

We now examine in detail a particular policy search example in which we have a two-action MDP, and a linear policy class is used. This case is interesting because, if the term $\mathrm{E}_{s \sim \mu_t}[V_{\pi,\pi_{t+1},...,\pi_{T-1}}(s)]$ (from the maximization step in the algorithm) can be nearly maximized by some linear policy $\pi$, then a good approximation to $\pi$ can be found.

Let $A = \{a_1, a_2\}$, and $\Pi = \{\pi_\theta(s) : \theta \in \mathbb{R}^n\}$, where $\pi_\theta(s) = a_1$ if $\theta^T \phi(s) \geq 0$, and $\pi_\theta(s) = a_2$ otherwise. Here, $\phi(s) \in \mathbb{R}^n$ is a vector of features of the state $s$. Consider the maximization step in the PSDP algorithm. Letting $1\{\cdot\}$ be the indicator function $(1\{\mathrm{True}\} = 1, 1\{\mathrm{False}\} = 0)$, we have the following algorithm for performing the maximization:

**Algorithm 2 (Linear maximization)** *Given $m1$ and $m2$:*

> *for $i = 1$ to $m_1$*
>
>> *Sample $s^{(i)} \sim \mu_t$.*
>> *Use $m_2$ Monte Carlo samples to estimate $V_{a_1,\pi_{t+1},...,\pi_{T-1}}(s^{(i)})$ and $V_{a_2,\pi_{t+1},...,\pi_{T-1}}(s^{(i)})$. Call the resulting estimates $q_1$ and $q_2$.*
>> *Let $y^{(i)} = 1\{q_1 > q_2\}$, and $w^{(i)} = |q_1 - q_2|$.*
>
> *Find $\theta = \arg\min_\theta \sum_{i=1}^{m_1} w^{(i)} 1\{1\{\theta^T \phi(s^{(i)}) \geq 0\} \neq y^{(i)}\}$.*
>
> *Output $\pi_\theta$.*

Intuitively, the algorithm does the following: It samples $m_1$ states $s^{(1)}, \ldots, s^{(m_1)}$ from the distribution $\mu_t$. Using $m_2$ Monte Carlo samples, it determines if action $a_1$ or action $a_2$ is preferable from that state, and creates a "label" $y^{(i)}$ for that state accordingly. Finally, it tries to find a linear decision boundary separating the states from which $a_1$ is better from the states from which $a_2$ is better. Further, the "importance" or "weight" $w^{(i)}$ assigned to $s^{(i)}$ is proportional to the difference in the values of the two actions from that state.

The final maximization step can be approximated via a call to any standard supervised learning algorithm that tries to find linear decision boundaries, such as a support vector machine or logistic regression. In some of our experiments, we use a weighted logistic regression to perform this maximization. However, using linear programming, it is possible to approximate this maximization. Let

$$T(\theta) = \sum_{i=1}^{m_1} w^{(i)} 1\{1\{\theta^T \phi(s^{(i)}) \geq 0\} \neq y^{(i)}\}$$

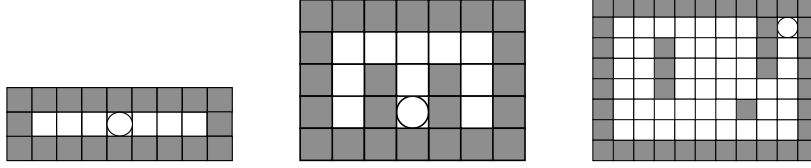

Figure 1: Illustrations of mazes: (a) Hallway (b) McCallum's Maze (c) Sutton's Maze

be the objective in the minimization. If there is a value of $\theta$ that can satisfies $T(\theta) = 0$, then it can be found via linear programming. Specifically, for each value of $i$, we let there be a constraint

$$\begin{cases} \theta^T \phi(s^{(i)}) > \kappa & \text{if } y^{(i)} = 1 \\ \theta^T \phi(s^{(i)}) < -\kappa & \text{otherwise} \end{cases}$$

otherwise, where $\kappa$ is any small positive constant. In the case in which these constraints cannot be simultaneously satisfied, it is NP-hard to find $\arg\min_\theta T(\theta)$. [1] However, the optimal value can be approximated. Specifically, if $\theta^* = \arg\min_\theta T(\theta)$, then [1] presents a polynomial time algorithm that finds $\theta$ so that

$$T(\theta) \leq (n+1)T(\theta^*).$$

Here, $n$ is the dimension of $\theta$. Therefore, if there is a linear policy that does well, we also find a policy that does well. (Conversely, if there is no linear policy that does well—i.e., if $T(\theta^*)$ above were large—then the bound would be very loose; however, in this setting there is no good linear policy, and hence we arguably should not be using a linear policy anyway or should consider adding more features.)

## 5    Experiments

The experiments below demonstrate each of the instantiations described previously.

### 5.1    POMDP gridworld example

Here we apply PSDP to some simple maze POMDPs (Figure (5.1)) to demonstrate its performance. In each the robot can move in any of the 4 cardinal direction. Except in (5.1c), the observation at each grid-cell is simply the directions in which the robot can freely move. The goal in each is to reach the circled grid cell in the minimum total number of steps from each starting cell.

First we consider the hallway maze in Figure (5.1a). The robot here is confounded by all the middle states appearing the same, and the optimal stochastic policy must take time at least quadratic in the length of the hallway to ensure it gets to the goal from both sides. PSDP deduces a non-stationary deterministic policy with much better performance: first clear the left half maze by always traveling right and then the right half maze by always traveling left.

McCallum's maze (Figure 5.1b) is discussed in the literature as admitting no satisficing deterministic reactive policy. When one allows non-stationary policies, however, solutions do exist: PSDP provides a policy with 55 total steps to goal. In our final benchmark, Sutton's maze (Figure 5.1c), the observations are determined by the openness of all eight connected directions.

Below we summarize the total number of steps to goal of our algorithm as compared with optimality for two classes of policy. Column 1 denotes PSDP performance using a uniform baseline distribution. The next column lists the performance of iterating PSDP, starting initially with a uniform baseline $\mu$ and then computing with a new baseline $\mu'$ based on the previously constructed policy. [2] Column 3 corresponds to optimal stationary deterministic

policy while the final column gives the best theoretically achievable performance given arbitrary memory. It is worthwhile to note that the PSDP computations are very fast in all of these problems, taking well under a second in an interpreted language.

|          | $\mu$ uniform | $\mu$ iterated | Optimal SD | Optimal     |
| -------- | ------------- | -------------- | ---------- | ----------- |
| Hallway  | 21            | 21             | $\infty$   | 18          |
| McCallum | 55            | 48             | $\infty$   | 39          |
| Sutton   | 412           | 412            | 416        | $\geq 408$  |

## 5.2  Robot walking

Our work is related in spirit to Atkeson and Morimoto [2], which describes a differential dynamic programming (DDP) algorithm that learns quadratic value functions along trajectories. These trajectories, which serve as an analog of our $\mu$ distribution, are then refined using the resulting policies. A central difference is their use of the value function backups as opposed to policy backups. In tackling the control problem presented in [2] we demonstrate ways in which PSDP extends that work.

[2] considers a planar biped robot that walks along a bar. The robot has two legs and a motor that applies torque where they meet. As the robot lacks knees, it walks by essentially brachiating (upside-down); a simple mechanism grabs the bar as a foot swings into position. The robot (excluding the position horizontally along the bar) can be described in a 5 dimensional state space using angles and angular velocities from the foot grasping the bar. The control variable that needs to be determined is the hip-torque.

In [2], significant manual "cost-function engineering" or "shaping" of the rewards was used to achieve walking at fixed speed. Much of this is due to the limitations of differential dynamic programming in which cost functions must always be locally quadratic. This rules out natural cost functions that directly penalize, for example, falling. As this limitation does not apply to our algorithm, we used a cost function that rewards the robot for each time-step it remains upright. In addition, we penalize quadratically deviation from the nominal horizontal velocity of 0.4 m/s and control effort applied.

Samples of $\mu$ are generated in the same way [2] generates initial trajectories, using a parametric policy search. For our policy we approximated the action-value function with a locally-weighted linear regression. PSDP's policy significantly improves performance over the parametric policy search; while both keep the robot walking we note that PSDP incurs 31% less cost per step.

DDP makes strong, perhaps unrealistic assumptions about the observability of state variables. PSDP, in contrast, can learn policies with limited observability. By hiding state variables from the algorithm, this control problem demonstrates PSDP's leveraging of non-stationarity and ability to cope with partial observability. PSDP can make the robot walk without *any* observations; open loop control is sufficient to propel the robot, albeit at a significant reduction in performance and robustness. In Figure (5.2) we see the signal generated by the learned open-loop controller. This complex torque signal would be identical for arbitrary initial conditions— modulo sign-reversals, as the applied torque at the hip is inverted from the control signal whenever the stance foot is switched.

## 5.3  Double-pole balancing

Our third problem, double pole balancing, is similar to the standard inverted pendulum problem, except that two unactuated poles, rather than a single one, are attached to the cart, and it is our task to simultaneously keep both of them balanced. This makes the task significantly harder than the standard single pole problem.

Using the simulator provided by [3], we implemented PSDP for this problem. The state variables were the cart position $x$; cart velocity $\dot{x}$; the two poles' angles $\phi_1$ and $\phi_2$; and the poles' angular velocities $\dot{\phi}_1$ and $\dot{\phi}_2$. The two actions are to accelerate left

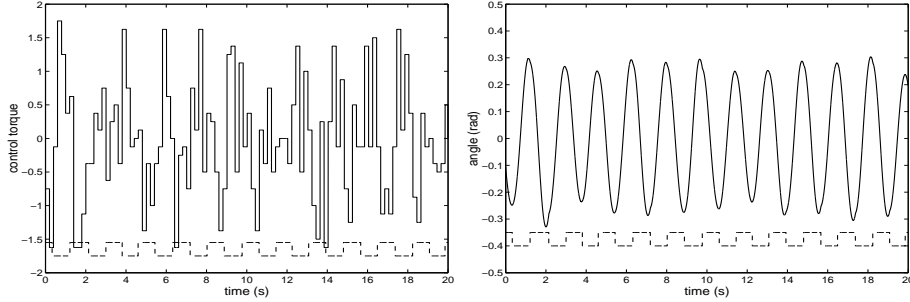

Figure 2: (Left) Control signal from open-loop learned controller. (Right) Resulting angle of one leg. The dashed line in each indicates which foot is grasping the bar at each time.

and to accelerate right. We used a linear policy class $\Pi$ as described previously, and $\phi(s) = [x, \dot{x}, \phi_1, \dot{\phi}_1, \phi_2, \dot{\phi}_2]^T$. By symmetry of the problem, a constant intercept term was unnecessary; leaving out an intercept enforces that if $a_1$ is the better action for some state $s$, then $a_2$ should be taken in the state $-s$.

The algorithm we used for the optimization step was logistic regression.[3] The baseline distribution $\mu$ that we chose was a zero-mean multivariate Gaussian distribution over all the state variables. Using a horizon of $T = 2000$ steps and 5000 Monte Carlo samples per iteration of the PSDP algorithm, we are able to successfully balance both poles.

**Acknowledgments.** We thank Chris Atkeson and John Langford for helpful conversations. J. Bagnell is supported by an NSF graduate fellowship. This work was also supported by NASA, and by the Department of the Interior/DARPA under contract number NBCH1020014.

## Footnotes

[1]If $S$ is continuous and $\mu_t$ and $\mu'_t$ are densities, the inner summation is replaced by an integral.

[2]It can be shown that this procedure of refining $\mu$ based on previous learned policies will never decrease performance.

[3]In our setting, we use weighted logistic regression and minimize $-\ell(\theta) = -\sum_i w^{(i)} \log p(y^{(i)}|s^{(i)}, \theta)$ where $p(y = 1|s, \theta) = 1/(1 + \exp(-\theta^T s))$. It is straightforward to show that this is a (convex) upper-bound on the objective function $T(\theta)$.

# References

[1] E. Amaldi and V. Kann. On the approximability of minimizing nonzero variables or unsatisfied relations in linear systems. *Theoretical Comp. Sci.*, 1998.

[2] C. Atkeson and J. Morimoto. Non-parametric representation of a policies and value functions: A trajectory based approach. In *NIPS 15*, 2003.

[3] F. Gomez.
http://www.cs.utexas.edu/users/nn/pages/software/software.html.

[4] Sham Kakade. *On the Sample Complexity of Reinforcement Learning*. PhD thesis, University College London, 2003.

[5] Sham Kakade and John Langford. Approximately optimal approximate reinforcement learning. In *Proc. 19th International Conference on Machine Learning*, 2002.

[6] Michael Kearns, Yishay Mansour, and Andrew Y. Ng. Approximate planning in large POMDPs via reusable trajectories. *(extended version of paper in NIPS 12)*, 1999.

[7] M. Littman. Memoryless policies: theoretical limitations and practical results. In *Proc. 3rd Conference on Simulation of Adaptive Behavior*, 1994.

[8] Andrew Y. Ng and Michael I. Jordan. PEGASUS: A policy search method for large MDPs and POMDPs. In *Proc. 16th Conf. Uncertainty in Artificial Intelligence*, 2000.

[9] Ronald J. Williams. Simple statistical gradient-following algorithms for connectionist reinforcement learning. *Machine Learning*, 8:229–256, 1992.

